# Using Deep Belief Nets to Learn Covariance Kernels for Gaussian Processes

**Ruslan Salakhutdinov and Geoffrey Hinton**
Department of Computer Science, University of Toronto
6 King's College Rd, M5S 3G4, Canada
rsalakhu,hinton@cs.toronto.edu

## Abstract

We show how to use unlabeled data and a deep belief net (DBN) to learn a good covariance kernel for a Gaussian process. We first learn a deep generative model of the unlabeled data using the fast, greedy algorithm introduced by [7]. If the data is high-dimensional and highly-structured, a Gaussian kernel applied to the top layer of features in the DBN works much better than a similar kernel applied to the raw input. Performance at both regression and classification can then be further improved by using backpropagation through the DBN to discriminatively fine-tune the covariance kernel.

## 1  Introduction

Gaussian processes (GP's) are a widely used method for Bayesian non-linear non-parametric regression and classification [13, 16]. GP's are based on defining a similarity or kernel function that encodes prior knowledge of the smoothness of the underlying process that is being modeled. Because of their flexibility and computational simplicity, GP's have been successfully used in many areas of machine learning.

Many real-world applications are characterized by high-dimensional, highly-structured data with a large supply of unlabeled data but a very limited supply of labeled data. Applications such as information retrieval and machine vision are examples where unlabeled data is readily available. GP's are discriminative models by nature and within the standard regression or classification scenario, unlabeled data is of no use. Given a set of *i.i.d.* labeled input vectors $\mathbf{X}_l = \{\mathbf{x}_n\}_{n=1}^N$ and their associated target labels $\{y_n\}_{n=1}^N \in R$ or $\{y_n\}_{n=1}^N \in \{-1, 1\}$ for regression/classification, GP's model $p(y_n|\mathbf{x}_n)$ directly. Unless some assumptions are made about the underlying distribution of the input data $\mathbf{X} = [\mathbf{X}_l, \mathbf{X}_u]$, unlabeled data, $\mathbf{X}_u$, cannot be used. Many researchers have tried to use unlabeled data by incorporating a model of $p(\mathbf{X})$. For classification tasks, [11] model $p(\mathbf{X})$ as a mixture $\sum_{y_n} p(x_n|y_n)p(y_n)$ and then infer $p(y_n|x_n)$, [15] attempts to learn covariance kernels based on $p(\mathbf{X})$, and [10] assumes that the decision boundaries should occur in regions where the data density, $p(\mathbf{X})$, is low. When faced with high-dimensional, highly-structured data, however, none of the existing approaches have proved to be particularly successful.

In this paper we exploit two properties of DBN's. First, they can be learned efficiently from unlabeled data and the top-level features generally capture significant, high-order correlations in the data. Second, they can be discriminatively fine-tuned using backpropagation. We first learn a DBN model of $p(\mathbf{X})$ in an entirely unsupervised way using the fast, greedy learning algorithm introduced by [7] and further investigated in [2, 14, 6]. We then use this generative model to initialize a multi-layer, non-linear mapping $F(\mathbf{x}|W)$, parameterized by $W$, with $F : \mathbf{X} \rightarrow \mathbf{Z}$ mapping the input vectors in $\mathbf{X}$ into a feature space $\mathbf{Z}$. Typically the mapping $F(\mathbf{x}|W)$ will contain millions of parameters. The top-level features produced by this mapping allow fairly accurate reconstruction of the input, so they must contain most of the information in the input vector, but they express this information in a way that makes explicit a lot of the higher-order structure in the input data.

After learning $F(\mathbf{x}|W)$, a natural way to define a kernel function is to set $K(\mathbf{x}_i, \mathbf{x}_j) = \exp\left(-||F(\mathbf{x}_i|W) - F(\mathbf{x}_j|W)||^2\right)$. Note that the kernel is initialized in an entirely unsupervised way. The parameters $W$ of the covariance kernel can then be fine-tuned using the labeled data by

maximizing the log probability of the labels with respect to $W$. In the final model most of the information for learning a covariance kernel will have come from modeling the input data. The very limited information in the labels will be used only to slightly adjust the layers of features already discovered by the DBN.

## 2  Gaussian Processes for Regression and Binary Classification

For a regression task, we are given a data set $\mathcal{D}$ of $i.i.d.$ labeled input vectors $\mathbf{X}_l = \{\mathbf{x}_n\}_{n=1}^N$ and their corresponding target labels $\{y_n\}_{n=1}^N \in R$. We are interested in the following probabilistic regression model:

$$y_n = f(x_n) + \epsilon, \qquad \epsilon \sim \mathcal{N}(\epsilon|0, \sigma^2) \tag{1}$$

A Gaussian process regression places a zero-mean GP prior over the underlying latent function $\mathbf{f}$ we are modeling, so that a-priori $p(\mathbf{f}|\mathbf{X}_l) = \mathcal{N}(\mathbf{f}|0, K)$, where $\mathbf{f} = [f(x_1), ..., f(x_n)]^T$ and $K$ is the covariance matrix, whose entries are specified by the covariance function $K_{ij} = K(\mathbf{x}_i, \mathbf{x}_j)$. The covariance function encodes our prior notion of the smoothness of $\mathbf{f}$, or the prior assumption that if two input vectors are similar according to some distance measure, their labels should be highly correlated. In this paper we will use the spherical Gaussian kernel, parameterized by $\theta = \{\alpha, \beta\}$:

$$K_{ij} = \alpha \exp\big(-\frac{1}{2\beta}(\mathbf{x}_i - \mathbf{x}_j)^T(\mathbf{x}_i - \mathbf{x}_j)\big) \tag{2}$$

Integrating out the function values $\mathbf{f}$, the marginal log-likelihood takes form:

$$L = \log p(\mathbf{y}|\mathbf{X}_l) = -\frac{N}{2}\log 2\pi - \frac{1}{2}\log|K + \sigma^2 I| - \frac{1}{2}\mathbf{y}^T(K + \sigma^2 I)^{-1}\mathbf{y} \tag{3}$$

which can then be maximized with respect to the parameters $\theta$ and $\sigma$. Given a new test point $\mathbf{x}_*$, a prediction is obtained by conditioning on the observed data and $\theta$. The distribution of the predicted value $y_*$ at $\mathbf{x}_*$ takes the form:

$$p(y_*|\mathbf{x}_*, \mathcal{D}, \theta, \sigma^2) = \mathcal{N}(y_*|k_*^T(K + \sigma^2 I)^{-1}\mathbf{y}, k_{**} - k_*^T(K + \sigma^2 I)^{-1}k_* + \sigma^2) \tag{4}$$

where $k_* = K(\mathbf{x}_*, \mathbf{X}_l)$, and $k_{**} = K(\mathbf{x}_*, \mathbf{x}_*)$.

For a binary classification task, we similarly place a zero mean GP prior over the underlying latent function $\mathbf{f}$, which is then passed through the logistic function $g(x) = 1/(1 + \exp(-x))$ to define a prior $p(y_n = 1|\mathbf{x}_n) = g(f(\mathbf{x}_n))$. Given a new test point $\mathbf{x}_*$, inference is done by first obtaining the distribution over the latent function $f_* = f(\mathbf{x}_*)$:

$$p(f_*|x_*, \mathcal{D}) = \int p(f_*|\mathbf{x}_*, \mathbf{X}_l, \mathbf{f})p(\mathbf{f}|\mathbf{X}_l, \mathbf{y})d\mathbf{f} \tag{5}$$

which is then used to produce a probabilistic prediction:

$$p(y_* = 1|\mathbf{x}_*, \mathcal{D}) = \int g(f_*)p(f_*|x_*, \mathcal{D})df_* \tag{6}$$

The non-Gaussian likelihood makes the integral in Eq. 5 analytically intractable. In our experiments, we approximate the non-Gaussian posterior $p(\mathbf{f}|\mathbf{X}_l, \mathbf{y})$ with a Gaussian one using expectation propagation [12]. For more thorough reviews and implementation details refer to [13, 16].

## 3  Learning Deep Belief Networks (DBN's)

In this section we describe an unsupervised way of learning a DBN model of the input data $\mathbf{X} = [\mathbf{X}_l, \mathbf{X}_u]$, that contains both labeled and unlabeled data sets. A DBN can be trained efficiently by using a Restricted Boltzmann Machine (RBM) to learn one layer of hidden features at a time [7]. Welling $et.\ al.$ [18] introduced a class of two-layer undirected graphical models that generalize RBM's to exponential family distributions. This framework will allow us to model real-valued images of face patches and word-count vectors of documents.

### 3.1  Modeling Real-valued Data

We use a conditional Gaussian distribution for modeling observed "visible" pixel values $\mathbf{x}$ (e.g. images of faces) and a conditional Bernoulli distribution for modeling "hidden" features $\mathbf{h}$ (Fig. 1):

$$p(x_i = x|\mathbf{h}) = \frac{1}{\sqrt{2\pi}\sigma_i}\exp(-\frac{(x - b_i - \sigma_i \sum_j h_j w_{ij})^2}{2\sigma_i^2}) \tag{7}$$

$$p(h_j = 1|\mathbf{x}) = g\big(b_j + \sum_i w_{ij}\frac{x_i}{\sigma_i}\big) \tag{8}$$

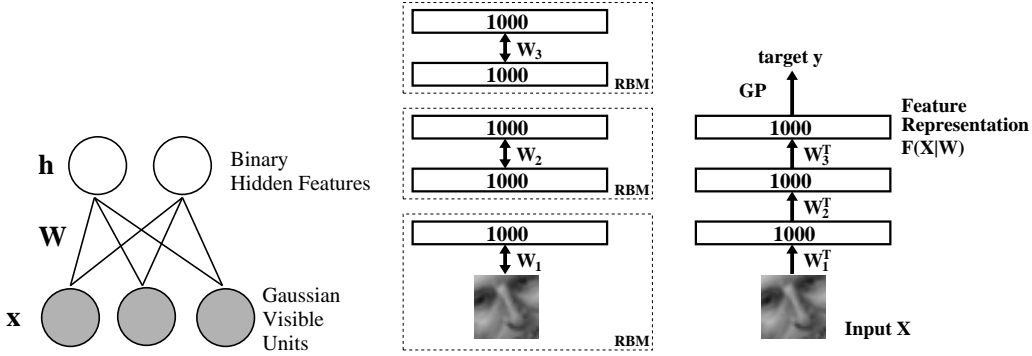

Figure 1: Left panel: Markov random field of the generalized RBM. The top layer represents stochastic binary hidden features **h** and and the bottom layer is composed of linear visible units **x** with Gaussian noise. When using a Constrained Poisson Model, the top layer represents stochastic binary latent topic features **h** and the bottom layer represents the Poisson visible word-count vector **x**. Middle panel: Pretraining consists of learning a stack of RBM's. Right panel: After pretraining, the RBM's are used to initialize a covariance function of the Gaussian process, which is then fine-tuned by backpropagation.

where $g(x) = 1/(1 + \exp(-x))$ is the logistic function, $w_{ij}$ is a symmetric interaction term between input $i$ and feature $j$, $\sigma_i^2$ is the variance of input $i$, and $b_i$, $b_j$ are biases. The marginal distribution over visible vector **x** is:

$$p(\mathbf{x}) = \sum_{\mathbf{h}} \frac{\exp\left(-E(\mathbf{x}, \mathbf{h})\right)}{\int_{\mathbf{u}} \sum_{\mathbf{g}} \exp\left(-E(\mathbf{u}, \mathbf{g})\right) d\mathbf{u}} \tag{9}$$

where $E(\mathbf{x}, \mathbf{h})$ is an energy term: $E(\mathbf{x}, \mathbf{h}) = \sum_i \frac{(x_i - b_i)^2}{2\sigma_i^2} - \sum_j b_j h_j - \sum_{i,j} h_j w_{ij} \frac{x_i}{\sigma_i}$. The parameter updates required to perform gradient ascent in the log-likelihood is obtained from Eq. 9:

$$\Delta w_{ij} = \epsilon \frac{\partial \log p(\mathbf{x})}{\partial w_{ij}} = \epsilon(<z_i h_j>_{data} - <z_i h_j>_{model}) \tag{10}$$

where $\epsilon$ is the learning rate, $z_i = x_i/\sigma_i$, $<\cdot>_{data}$ denotes an expectation with respect to the data distribution and $<\cdot>_{model}$ is an expectation with respect to the distribution defined by the model. To circumvent the difficulty of computing $<\cdot>_{model}$, we use 1-step Contrastive Divergence [5]:

$$\Delta w_{ij} = \epsilon(<z_i h_j>_{data} - <z_i h_j>_{recon}) \tag{11}$$

The expectation $<z_i h_j>_{data}$ defines the expected sufficient statistics of the data distribution and is computed as $z_i p(h_j = 1|\mathbf{x})$ when the features are being driven by the observed data from the training set using Eq. 8. After stochastically activating the features, Eq. 7 is used to "reconstruct" real-valued data. Then Eq. 8 is used again to activate the features and compute $<z_i h_j>_{recon}$ when the features are being driven by the reconstructed data. Throughout our experiments we set variances $\sigma_i^2 = 1$ for all visible units $i$, which facilitates learning. The learning rule for the biases is just a simplified version of Eq. 11.

### 3.2 Modeling Count Data with the Constrained Poisson Model

We use a conditional "constrained" Poisson distribution for modeling observed "visible" word count data **x** and a conditional Bernoulli distribution for modeling "hidden" topic features **h**:

$$p(x_i = n|\mathbf{h}) = \text{Pois}\left(n, \frac{\exp\left(\lambda_i + \sum_j h_j w_{ij}\right)}{\sum_k \exp\left(\lambda_k + \sum_j h_j W_{kj}\right)} \times N\right), \quad p(h_j = 1|\mathbf{x}) = g(b_j + \sum_i w_{ij} x_i) \tag{12}$$

where $\text{Pois}(n, \lambda) = e^{-\lambda} \lambda^n / n!$, $w_{ij}$ is a symmetric interaction term between word $i$ and feature $j$, $N = \sum_i x_i$ is the total length of the document, $\lambda_i$ is the bias of the conditional Poisson model for word $i$, and $b_j$ is the bias of feature $j$. The Poisson rate, whose log is shifted by the weighted combination of the feature activations, is normalized and scaled up by $N$. We call this the "Constrained Poisson Model" since it ensures that the mean Poisson rates across all words sum up to the length of the document. This normalization is significant because it makes learning stable and it deals appropriately with documents of different lengths.

The marginal distribution over visible count vectors $\mathbf{x}$ is given in Eq. 9 with an "energy" given by

$$E(\mathbf{x}, \mathbf{h}) = -\sum_i \lambda_i x_i + \sum_i \log(x_i!) - \sum_j b_j h_j - \sum_{i,j} x_i h_j w_{ij} \tag{13}$$

The gradient of the log-likelihood function is:

$$\Delta w_{ij} = \epsilon \frac{\partial \log p(\mathbf{v})}{\partial w_{ij}} = \epsilon(<x_i h_j>_{data} - <x_i h_j>_{model}) \tag{14}$$

### 3.3 Greedy Recursive Learning of Deep Belief Nets

A single layer of binary features is not the best way to capture the structure in the input data. We now describe an efficient way to learn additional layers of binary features.

After learning the first layer of hidden features we have an undirected model that defines $p(\mathbf{v}, \mathbf{h})$ by defining a consistent pair of conditional probabilities, $p(\mathbf{h}|\mathbf{v})$ and $p(\mathbf{v}|\mathbf{h})$ which can be used to sample from the model distribution. A different way to express what has been learned is $p(\mathbf{v}|\mathbf{h})$ and $p(\mathbf{h})$. Unlike a standard, directed model, this $p(\mathbf{h})$ does not have its own separate parameters. It is a complicated, non-factorial prior on $\mathbf{h}$ that is defined implicitly by $p(\mathbf{h}|\mathbf{v})$ and $p(\mathbf{v}|\mathbf{h})$. This peculiar decomposition into $p(\mathbf{h})$ and $p(\mathbf{v}|\mathbf{h})$ suggests a recursive algorithm: keep the learned $p(\mathbf{v}|\mathbf{h})$ but replace $p(\mathbf{h})$ by a better prior over $\mathbf{h}$, *i.e.* a prior that is closer to the average, over all the data vectors, of the conditional posterior over $\mathbf{h}$. So after learning an undirected model, the part we keep is part of a multilayer *directed* model.

We can sample from this average conditional posterior by simply using $p(\mathbf{h}|\mathbf{v})$ on the training data and these samples are then the "data" that is used for training the next layer of features. The only difference from learning the first layer of features is that the "visible" units of the second-level RBM are also binary [6, 3]. The learning rule provided in the previous section remains the same [5]. We could initialize the new RBM model by simply using the existing learned model but with the roles of the hidden and visible units reversed. This ensures that $p(\mathbf{v})$ in our new model starts out being exactly the same as $p(\mathbf{h})$ in our old one. Provided the number of features per layer does not decrease, [7] show that each extra layer increases a variational lower bound on the log probability of data. To suppress noise in the learning signal, we use the real-valued activation *probabilities* for the visible units of every RBM, but to prevent hidden units from transmitting more than one bit of information from the data to its reconstruction, the pretraining always uses stochastic binary values for the hidden units.

The greedy, layer-by-layer training can be repeated several times to learn a deep, hierarchical model in which each layer of features captures strong high-order correlations between the activities of features in the layer below.

## 4   Learning the Covariance Kernel for a Gaussian Process

After pretraining, the stochastic activities of the binary features in each layer are replaced by deterministic, real-valued probabilities and the DBN is used to initialize a multi-layer, non-linear mapping $f(\mathbf{x}|W)$ as shown in figure 1. We define a Gaussian covariance function, parameterized by $\theta = \{\alpha, \beta\}$ and $W$ as:

$$K_{ij} = \alpha \exp\left(-\frac{1}{2\beta}||F(\mathbf{x}_i|W) - F(\mathbf{x}_j|W)||^2\right) \tag{15}$$

Note that this covariance function is initialized in an entirely unsupervised way. We can now maximize the log-likelihood of Eq. 3 with respect to the parameters of the covariance function using the labeled training data[9]. The derivative of the log-likelihood with respect to the kernel function is:

$$\frac{\partial L}{\partial K_y} = \frac{1}{2}\left(K_y^{-1}\mathbf{y}\mathbf{y}^T K_y^{-1} - K_y^{-1}\right) \tag{16}$$

where $K_y = K + \sigma^2 I$ is the covariance matrix. Using the chain rule we readily obtain the necessary gradients:

$$\frac{\partial L}{\partial \theta} = \frac{\partial L}{\partial K_y}\frac{\partial K_y}{\partial \theta} \quad \text{and} \quad \frac{\partial L}{W} = \frac{\partial L}{\partial K_y}\frac{\partial K_y}{\partial F(\mathbf{x}|W)}\frac{\partial F(\mathbf{x}|W)}{\partial W} \tag{17}$$

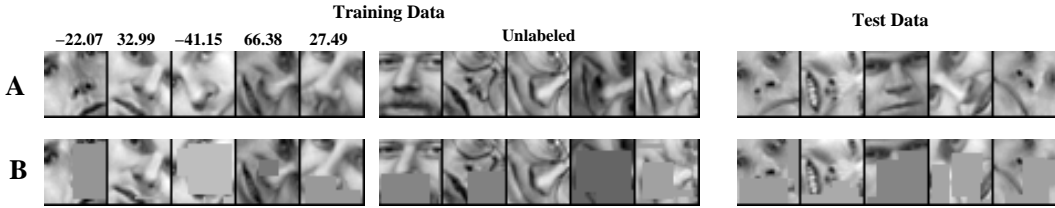

Figure 2: Top panel **A**: Randomly sampled examples of the training and test data. Bottom panel **B**: The same sample of the training and test images but with rectangular occlusions.

|   | Training labels | **GPstandard** | | **GP-DBNgreedy** | | **GP-DBNfine** | | **GPpca** | |
|---|---|---|---|---|---|---|---|---|---|
|   |   | Sph. | ARD | Sph. | ARD | Sph. | ARD | Sph. | ARD |
| A | 100 | 22.24 | 28.57 | 17.94 | 18.37 | 15.28 | **15.01** | 18.13 (10) | 16.47 (10) |
|   | 500 | 17.25 | 18.16 | 12.71 | 8.96 | 7.25 | **6.84** | 14.75 (20) | 10.53 (80) |
|   | 1000 | 16.33 | 16.36 | 11.22 | 8.77 | 6.42 | **6.31** | 14.86 (20) | 10.00 (160) |
| B | 100 | 26.94 | 28.32 | 23.15 | 19.42 | 19.75 | **18.59** | 25.91 (10) | 19.27 (20) |
|   | 500 | 20.20 | 21.06 | 15.16 | 11.01 | 10.56 | **10.12** | 17.67 (10) | 14.11 (20) |
|   | 1000 | 19.20 | 17.98 | 14.15 | 10.43 | **9.13** | 9.23 | 16.26 (10) | 11.55 (80) |

Table 1: Performance results on the face-orientation regression task. The root mean squared error (RMSE) on the test set is shown for each method using spherical Gaussian kernel and Gaussian kernel with ARD hyper-parameters. By row: A) Non-occluded face data, B) Occluded face data. For the GPpca model, the number of principal components that performs best on the test data is shown in parenthesis.

where $\partial F(\mathbf{x}|W)/\partial W$ is computed using standard backpropagation. We also optimize the observation noise $\sigma^2$. It is necessary to compute the inverse of $K_y$, so each gradient evaluation has $O(N^3)$ complexity where $N$ is the number of the labeled training cases. When learning the restricted Boltzmann machines that are composed to form the initial DBN, however, each gradient evaluation scales linearly in time and space with the number of unlabeled training cases. So the pretraining stage can make efficient use of very large sets of unlabeled data to create sensible, high-level features and when the amount of labeled data is small. Then the very limited amount of information in the labels can be used to slightly refine those features rather than to create them.

## 5 Experimental Results

In this section we present experimental results for several regression and classification tasks that involve high-dimensional, highly-structured data. The first regression task is to extract the orientation of a face from a gray-level image of a large patch of the face. The second regression task is to map images of handwritten digits to a single real-value that is as close as possible to the integer represented by the digit in the image. The first classification task is to discriminate between images of odd digits and images of even digits. The second classification task is to discriminate between two different classes of news story based on the vector of word counts in each story.

### 5.1 Extracting the Orientation of a Face Patch

The Olivetti face data set contains ten 64×64 images of each of forty different people. We constructed a data set of 13,000 28×28 images by randomly rotating ($-90°$ to $+90°$), cropping, and subsampling the original 400 images. The data set was then subdivided into 12,000 training images, which contained the first 30 people, and 1,000 test images, which contained the remaining 10 people. 1,000 randomly sampled face patches from the training set were assigned an orientation label. The remaining 11,000 training images were used as unlabeled data. We also made a more difficult version of the task by occluding part of each face patch with randomly chosen rectangles. Panel A of figure 2 shows randomly sampled examples from the training and test data.

For training on the Olivetti face patches we used the 784-1000-1000-1000 architecture shown in figure 1. The entire training set of 12,000 unlabeled images was used for greedy, layer-by-layer training of a DBN model. The 2.8 million parameters of the DBN model may seem excessive for 12,000 training cases, but each training case involves modeling 625 real-values rather than just a single real-valued label. Also, we only train each layer of features for a few passes through the training data and we penalize the squared weights.

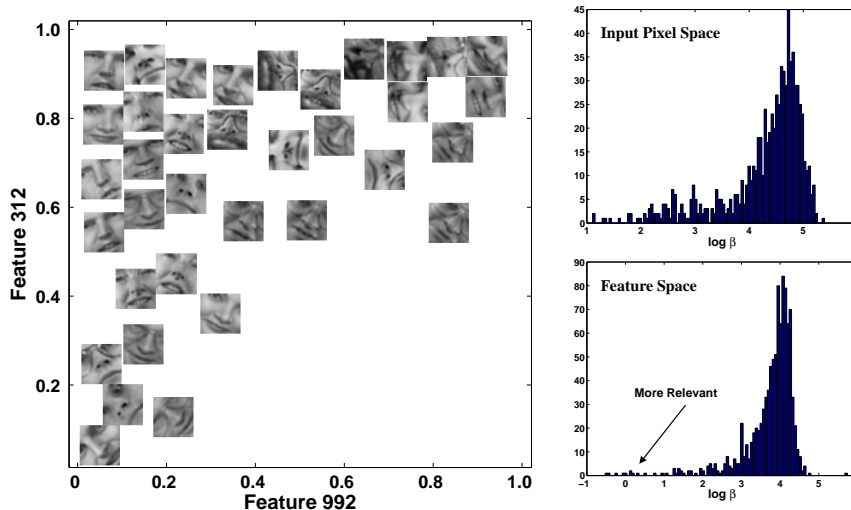

Figure 3: Left panel shows a scatter plot of the two most relevant features, with each point replaced by the corresponding input test image. For better visualization, overlapped images are not shown. Right panel displays the histogram plots of the learned ARD hyper-parameters $\log \beta$.

After the DBN has been pretrained on the unlabeled data, a GP model was fitted to the labeled data using the top-level features of the DBN model as inputs. We call this model **GP-DBNgreedy**. GP-DBNgreedy can be significantly improved by slightly altering the weights in the DBN. The GP model gives error derivatives for its input vectors which are the top-level features of the DBN. These derivatives can be backpropagated through the DBN to allow discriminative fine-tuning of the weights. Each time the weights in the DBN are updated, the GP model is also refitted. We call this model **GP-DBNfine**. For comparison, we fitted a GP model that used the pixel intensities of the labeled images as its inputs. We call this model **GPstandard**. We also used PCA to reduce the dimensionality of the labeled images and fitted several different GP models using the projections onto the first $m$ principal components as the input. Since we only want a lower bound on the error of this model, we simply use the value of $m$ that performs best on the *test* data. We call this model **GPpca**. Table 1 shows the root mean squared error (RMSE) of the predicted face orientations using all four types of GP model on varying amounts of labeled data. The results show that both GP-DBNgreedy and GP-DBNfine significantly outperform a regular GP model. Indeed, GP-DBNfine with only 100 labeled training cases outperforms GPstandard with 1000.

To test the robustness of our approach to noise in the input we took the same data set and created artificial rectangular occlusions (see Fig. 2, panel B). The number of rectangles per image was drawn from a Poisson with $\lambda = 2$. The top-left location, length and width of each rectangle was sampled from a uniform [0,25]. The pixel intensity of each occluding rectangle was set to the mean pixel intensity of the entire image. Table 1 shows that the performance of all models degrades, but their relative performances remain the same and GP-DBNfine on occluded data is still much better than GPstandard on non-occluded data.

We have also experimented with using a Gaussian kernel with ARD hyper-parameters, which is a common practice when the input vectors are high-dimensional:

$$K_{ij} = \alpha \exp\left(-\frac{1}{2}(\mathbf{x}_i - \mathbf{x}_j)^T \mathbf{D}(\mathbf{x}_i - \mathbf{x}_j)\right) \tag{18}$$

where $\mathbf{D}$ is the diagonal matrix with $\mathbf{D}_{ii} = 1/\beta_i$, so that the covariance function has a separate length-scale parameter for each dimension. ARD hyper-parameters were optimized by maximizing the marginal log-likelihood of Eq. 3. Table 1 shows that ARD hyper-parameters do not improve GPstandard, but they do slightly improve GP-DBNfine and they strongly improve GP-DBNgreedy and GPpca when there are 500 or 1000 labeled training cases.

The histogram plot of $\log \beta$ in figure 3 reveals that there are a few extracted features that are very relevant (small $\beta$) to our prediction task. The same figure (left panel) shows a scatter plot of the two most relevant features of GP-DBNgreedy model, with each point replaced by the corresponding input test image. Clearly, these two features carry a lot of information about the orientation of the face.

| | Train labels | GPstandard | | GP-DBNgreedy | | GP-DBNfine | | GPpca | |
|---|---|---|---|---|---|---|---|---|---|
| | | Sph. | ARD | Sph. | ARD | Sph. | ARD | Sph. | ARD |
| A | 100 | 1.86 | 2.27 | 1.68 | 1.61 | 1.63 | **1.58** | 1.73 (20) | 2.00 (20) |
| | 500 | 1.42 | 1.62 | 1.19 | 1.27 | **1.16** | 1.22 | 1.32 (40) | 1.36 (20) |
| | 1000 | 1.25 | 1.36 | 1.07 | 1.14 | **1.03** | 1.10 | 1.19 (40) | 1.22 (80) |
| B | 100 | 0.0884 | 0.1087 | 0.0528 | 0.0597 | **0.0501** | 0.0599 | 0.0785 (10) | 0.0920 (10) |
| | 500 | 0.0222 | 0.0541 | 0.0100 | 0.0161 | **0.0055** | 0.0104 | 0.0160 (40) | 0.0235 (20) |
| | 1000 | 0.0129 | 0.0385 | 0.0058 | 0.0059 | **0.0050** | 0.0100 | 0.0091 (40) | 0.0127 (40) |

Table 2: Performance results on the digit magnitude regression task (A) and and discriminating odd vs. even digits classification task (B). The root mean squared error for regression task on the test set is shown for each method. For classification task the area under the ROC (AUROC) metric is used. For each method we show 1-AUROC on the test set. All methods were tried using both spherical Gaussian kernel, and a Gaussian kernel with ARD hyper-parameters. For the GPpca model, the number of principal components that performs best on the test data is shown in parenthesis.

| Number of labeled cases (50% in each class) | GPstandard | GP-DBNgreedy | GP-DBNfine |
|---|---|---|---|
| 100 | 0.1295 | 0.1180 | **0.0995** |
| 500 | 0.0875 | 0.0793 | **0.0609** |
| 1000 | 0.0645 | 0.0580 | **0.0458** |

Table 3: Performance results using the area under the ROC (AUROC) metric on the text classification task. For each method we show 1-AUROC on the test set.

We suspect that the GP-DBNfine model does not benefit as much from the ARD hyper-parameters because the fine-tuning stage is already capable of turning down the activities of irrelevant top-level features.

## 5.2 Extracting the Magnitude Represented by a Handwritten Digit and Discriminating between Images of Odd and Even Digits

The MNIST digit data set contains 60,000 training and 10,000 test $28 \times 28$ images of ten handwritten digits (0 to 9). 100 randomly sampled training images of each class were assigned a magnitude label. The remaining 59,000 training images were used as unlabeled data. As in the previous experiment, we used the 784-1000-1000-1000 architecture with the entire training set of 60,000 unlabeled digits being used for greedily pretraining the DBN model. Table 2, panel A, shows that GP-DBNfine and GP-DBNgreedy perform considerably better than GPstandard both with and without ARD hyper-parameters. The same table, panel B, shows results for the classification task of discriminating between images of odd and images of even digits. We used the same labeled training set, but with each digit categorized into an even or an odd class. The same DBN model was used, so the Gaussian covariance function was initialized in exactly the same way for both regression and classification tasks. The performance of GP-DBNgreedy demonstrates that the greedily learned feature representation captures a lot of structure in the unlabeled input data which is useful for subsequent discrimination tasks, even though these tasks are unknown when the DBN is being trained.

## 5.3 Classifying News Stories

The Reuters Corpus Volume II is an archive of 804,414 newswire stories The corpus covers four major groups: Corporate/Industrial, Economics, Government/Social, and Markets. The data was randomly split into 802,414 training and 2000 test articles. The test set contains 500 articles of each major group. The available data was already in a convenient, preprocessed format, where common stopwords were removed and all the remaining words were stemmed. We only made use of the 2000 most frequently used word stems in the training data. As a result, each document was represented as a vector containing 2000 word counts. No other preprocessing was done.

For the text classification task we used a 2000-1000-1000-1000 architecture. The entire unlabeled training set of 802,414 articles was used for learning a multilayer generative model of the text documents. The bottom layer of the DBN was trained using a Constrained Poisson Model. Table 3 shows the area under the ROC curve for classifying documents belonging to the Corporate/Industrial vs. Economics groups. As expected, GP-DBNfine and GP-DBNgreedy work better than GPstandard. The results of binary discrimination between other pairs of document classes are very similar to the results presented in table 3. Our experiments using a Gaussian kernel with ARD hyper-parameters did not show any significant improvements. Examining the histograms of the length-scale parame-

ters $\beta$, we found that most of the input word-counts as well as most of the extracted features were relevant to the classification task.

## 6    Conclusions and Future Research

In this paper we have shown how to use Deep Belief Networks to greedily pretrain and discriminatively fine-tune a covariance kernel for a Gaussian Process. The discriminative fine-tuning produces an additional improvement in performance that is comparable in magnitude to the improvement produced by using the greedily pretrained DBN. For high-dimensional, highly-structured data, this is an effective way to make use of large unlabeled data sets, especially when labeled training data is scarce. Greedily pretrained DBN's can also be used to provide input vectors for other kernel-based methods, including SVMs [17, 8] and kernel regression [1], and our future research will concentrate on comparing our method to other kernel-based semi-supervised learning algorithms [4, 19].

**Acknowledgments**

We thank Radford Neal for many helpful suggestions. This research was supported by NSERC, CFI and OTI. GEH is a fellow of CIAR and holds a CRC chair.

## References

[1] J. K. Benedetti. On the nonparametric estimation of regression functions. *Journal of the Royal Statistical Society series B*, 39:248–253, 1977.

[2] Y. Bengio and Y. Le Cun. Scaling learning algorithms towards AI. In L. Bottou, O. Chapelle, D. DeCoste, and J. Weston, editors, *Large-Scale Kernel Machines*. MIT Press, 2007.

[3] Y. Bengio, P. Lamblin, D. Popovici, and H. Larochelle. Greedy layer-wise training of deep networks. In *Advances in Neural Information Processing Systems*, 2006.

[4] O. Chapelle, B. Schölkopf, and A. Zien. *Semi-Supervised Learning*. MIT Press, 2006.

[5] G. E. Hinton. Training products of experts by minimizing contrastive divergence. *Neural Computation*, 14(8):1711–1800, 2002.

[6] G. E. Hinton and R. Salakhutdinov. Reducing the dimensionality of data with neural networks. *Science*, 313, 2006.

[7] Geoffrey E. Hinton, Simon Osindero, and Yee Whye Teh. A fast learning algorithm for deep belief nets. *Neural Computation*, 18(7):1527–1554, 2006.

[8] F. Lauer, C. Y. Suen, and G. Bloch. A trainable feature extractor for handwritten digit recognition. *Pattern Recognition*, 40(6):1816–1824, 2007.

[9] N. D. Lawrence and J. Quiñonero Candela. Local distance preservation in the GP-LVM through back constraints. In William W. Cohen and Andrew Moore, editors, *ICML*, volume 148, pages 513–520. ACM, 2006.

[10] N. D. Lawrence and M. I. Jordan. Semi-supervised learning via gaussian processes. In *NIPS*, 2004.

[11] N. D. Lawrence and B. Schölkopf. Estimating a kernel Fisher discriminant in the presence of label noise. In *Proc. 18th International Conf. on Machine Learning*, pages 306–313. Morgan Kaufmann, San Francisco, CA, 2001.

[12] T. P. Minka. Expectation propagation for approximate bayesian inference. In Jack Breese and Daphne Koller, editors, *UAI*, pages 362–369, San Francisco, CA, 2001. Morgan Kaufmann Publishers.

[13] C. E. Rasmussen and C. Williams. *Gaussian Processes for Machine Learning*. The MIT Press, 2006.

[14] R. Salakhutdinov and G. E. Hinton. Learning a nonlinear embedding by preserving class neighbourhood structure. In *AI and Statistics*, 2007.

[15] M. Seeger. Covariance kernels from bayesian generative models. In Thomas G. Dietterich, Suzanna Becker, and Zoubin Ghahramani, editors, *NIPS*, pages 905–912. MIT Press, 2001.

[16] M. Seeger. Gaussian processes for machine learning. *Int. J. Neural Syst*, 14(2):69–106, 2004.

[17] V. Vapnik. *Statistical Learning Theory*. Wiley, 1998.

[18] M. Welling, M. Rosen-Zvi, and G. Hinton. Exponential family harmoniums with an application to information retrieval. In *NIPS 17*, pages 1481–1488, Cambridge, MA, 2005. MIT Press.

[19] Xiaojin Zhu, Jaz S. Kandola, Zoubin Ghahramani, and John D. Lafferty. Nonparametric transforms of graph kernels for semi-supervised learning. In *NIPS*, 2004.

